# iLSTD: Eligibility Traces and Convergence Analysis

**Alborz Geramifard**　　　Michael Bowling　　　Martin Zinkevich
Richard S. Sutton
Department of Computing Science
University of Alberta
Edmonton, Alberta
{alborz,bowling,maz,sutton}@cs.ualberta.ca

## Abstract

We present new theoretical and empirical results with the iLSTD algorithm for policy evaluation in reinforcement learning with linear function approximation. iLSTD is an incremental method for achieving results similar to LSTD, the data-efficient, least-squares version of temporal difference learning, without incurring the full cost of the LSTD computation. LSTD is O($n^2$), where $n$ is the number of parameters in the linear function approximator, while iLSTD is O($n$). In this paper, we generalize the previous iLSTD algorithm and present three new results: (1) the first convergence proof for an iLSTD algorithm; (2) an extension to incorporate eligibility traces without changing the asymptotic computational complexity; and (3) the first empirical results with an iLSTD algorithm for a problem (mountain car) with feature vectors large enough ($n = 10,000$) to show substantial computational advantages over LSTD.

## 1  Introduction

A key part of many reinforcement learning algorithms is a policy evaluation process, in which the value function of a policy is estimated online from data. In this paper, we consider the problem of policy evaluation where the value function estimate is a linear function of state features and is updated after each time step.

Temporal difference (TD) learning is a common approach to this problem [Sutton, 1988]. The TD algorithm updates its value-function estimate based on the observed TD error on each time step. The TD update takes only $O(n)$ computation per time step, where $n$ is the number of features. However, because conventional TD methods do not make any later use of the time step's data, they may require a great deal of data to compute an accurate estimate. More recently, LSTD [Bradtke and Barto, 1996] and its extension LSTD($\lambda$) [Boyan, 2002] were introduced as alternatives. Rather than making updates on each step to improve the estimate, these methods maintain compact summaries of all observed state transitions and rewards and solve for the value function which has zero expected TD error over the observed data. However, although LSTD and LSTD($\lambda$) make more efficient use of the data, they require $O(n^2)$ computation per time step, which is often impractical for the large feature sets needed in many applications. Hence, practitioners are often faced with the dilemma of having to chose between excessive computational expense and excessive data expense.

Recently, Geramifard and colleagues [2006] introduced an *incremental* least-squares TD algorithm, *iLSTD*, as a compromise between the computational burden of LSTD and the relative data inefficiency of TD. The algorithm focuses on the common situation of large feature sets where only a small number of features are non-zero on any given time step. iLSTD's per-time-step computational complexity in this case is only $O(n)$. In empirical results on a simple problem, iLSTD exhibited a rate of learning similar to that of LSTD.

In this paper, we substantially extend the iLSTD algorithm, generalizing it in two key ways. First, we include the use of eligibility traces, defining iLSTD($\lambda$) consistent with the family of TD($\lambda$) and LSTD($\lambda$) algorithms. We show that, under the iLSTD assumptions, the per-time-step computational complexity of this algorithm remains linear in the number of features. Second, we generalize the feature selection mechanism. We prove that for a general class of selection mechanisms, iLSTD($\lambda$) converges to the same solution as TD($\lambda$) and LSTD($\lambda$), for all $0 \leq \lambda \leq 1$.

## 2 Background

Reinforcement learning is an approach to finding optimal policies in sequential decision making problems with an unknown environment [*e.g.*, see Sutton and Barto, 1998]. We focus on the class of environments known as Markov decision processes (MDPs). An MDP is a tuple, $(\mathcal{S}, \mathcal{A}, \mathcal{P}^a_{ss'}, \mathcal{R}^a_{ss'}, \gamma)$, where $\mathcal{S}$ is a set of states, $\mathcal{A}$ is a set of actions, $\mathcal{P}^a_{ss'}$ is the probability of reaching state $s'$ after taking action $a$ in state $s$, and $\mathcal{R}^a_{ss'}$ is the reward received when that transition occurs, and $\gamma \in [0, 1]$ is a discount rate parameter. A trajectory of experience is a sequence $s_0, a_0, r_1, s_1, a_1, r_2, s_2, \ldots$, where the agent in $s_1$ takes action $a_1$ and receives reward $r_2$ while transitioning to $s_2$ before taking $a_2$, *etc.*

Given a policy, one often wants to estimate the policy's state-value function, or expected sum of discounted future rewards:

$$V^\pi(s) \quad = \quad E\left[\sum_{t=1}^\infty \gamma^{t-1} r_t \middle| s_0 = s, \pi\right].$$

In particular, we are interested in approximating $V^\pi$ using a linear function approximator. Let $\phi : \mathcal{S} \to \Re^n$, be some features of the state space. Linear value functions are of the form

$$V_{\boldsymbol{\theta}}(s) = \phi(s)^T \boldsymbol{\theta},$$

where $\boldsymbol{\theta} \in \Re^n$ are the parameters of the value function. In this work we will exclusively consider *sparse feature representations*: for all states $s$ the number of non-zero features in $\phi(s)$ is no more than $k \ll n$. Sparse feature representations are quite common as a generic approach to handling non-linearity [*e.g.*, Stone *et al.*, 2005].[1]

### 2.1 Temporal Difference Learning

TD($\lambda$) is the traditional approach to policy evaluation [see Sutton and Barto, 1998]. It is based on the computation of a $\lambda$-*return*, $R^\lambda_t(V)$, at each time step:

$$R^\lambda_t(V) = (1 - \lambda) \sum_{k=1}^\infty \lambda^{k-1} \left(\gamma^k V(s_{t+k}) + \sum_{i=1}^k \gamma^{i-1} r_{t+i}\right).$$

Note that the $\lambda$-return is a weighted sum of $k$-step returns, each of which looks ahead $k$ steps summing the discounted rewards as well as the estimated value of the resulting state. The $\lambda$-return forms the basis of the update to the value function parameters:

$$\boldsymbol{\theta}_{t+1} = \boldsymbol{\theta}_t + \alpha_t \phi(s_t) \left(R_t(V_{\boldsymbol{\theta}_t}) - V_{\boldsymbol{\theta}_t}(s_t)\right),$$

where $\alpha_t$ is the learning rate. This "forward view" requires a complete trajectory to compute the $\lambda$-return and update the parameters. The "backward view" is a more efficient implementation that depends only on one-step returns and an eligibility trace vector:

$$
\begin{aligned}
\boldsymbol{\theta}_{t+1} &= \boldsymbol{\theta}_t + \alpha_t \mathbf{u}_t(\theta_t) \\
\mathbf{u}_t(\boldsymbol{\theta}) &= \mathbf{z}_t \left(r_{t+1} + \gamma V_{\boldsymbol{\theta}}(s_{t+1}) - V_{\boldsymbol{\theta}}(s_t)\right) \\
\mathbf{z}_t &= \lambda \gamma \mathbf{z}_{t+1} + \phi(s_t),
\end{aligned}
$$

where $\mathbf{z}_t$ is the eligibility trace and $\mathbf{u}_t(\boldsymbol{\theta})$ is the TD update. Notice that TD($\lambda$) requires only a constant number of vector operations and so is $O(n)$ per time step. In the special case where $\lambda = 0$ and the feature representation is sparse, this complexity can be reduced to $O(k)$. In addition, TD($\lambda$) is guaranteed to converge [Tsitsiklis and Van Roy, 1997].

## 2.2 Least-Squares TD

Least-squares TD (LSTD) was first introduced by Bradtke and Barto [1996] and later extended with $\lambda$-returns by Boyan [2002]. LSTD($\lambda$) can be viewed as immediately solving for the value function parameters which would result in the sum of TD updates over the observed trajectory being zero. Let $\boldsymbol{\mu}_t(\boldsymbol{\theta})$ be the sum of the TD updates through time $t$. If we let $\boldsymbol{\phi}_t = \boldsymbol{\phi}(s_t)$ then,

$$
\begin{aligned}
\boldsymbol{\mu}_t(\boldsymbol{\theta}) &= \sum_{i=1}^{t} \mathbf{u}_i(\boldsymbol{\theta}) = \sum_{i=1}^{t} \mathbf{z}_i \left( r_{i+1} + \gamma V_{\boldsymbol{\theta}}(s_{i+1}) - V_{\boldsymbol{\theta}}(s_i) \right) \\
&= \sum_{i=1}^{t} \mathbf{z}_i \left( r_{i+1} + \gamma \boldsymbol{\phi}_{i+1}^T \boldsymbol{\theta} - \boldsymbol{\phi}_i^T \boldsymbol{\theta} \right) \\
&= \underbrace{\sum_{i=1}^{t} \mathbf{z}_i r_{i+1}}_{\mathbf{b}_t} - \underbrace{\sum_{i=1}^{t} \mathbf{z}_i (\boldsymbol{\phi}_i - \gamma \boldsymbol{\phi}_{i+1})^T}_{\mathbf{A}_t} \boldsymbol{\theta} = \mathbf{b}_t - \mathbf{A}_t \boldsymbol{\theta}.
\end{aligned}
\tag{1}
$$

Since we want to choose parameters such that the sum of TD updates is zero, we set Equation 1 to zero and solve for the new parameter vector,

$$
\boldsymbol{\theta}_{t+1} = \mathbf{A}_t^{-1} \mathbf{b}_t.
$$

The online version of LSTD($\lambda$) incorporates each observed reward and state transition into the $\mathbf{b}$ vector and the $\mathbf{A}$ matrix and then solves for a new $\boldsymbol{\theta}$. Notice that, once $\mathbf{b}$ and $\mathbf{A}$ are updated, the experience tuple can be forgotten without losing any information. Because $\mathbf{A}$ only changes by a small amount on each time step, $\mathbf{A}^{-1}$ can also be maintained incrementally. The computation requirement is $O(n^2)$ per time step. Like TD($\lambda$), LSTD($\lambda$) is guaranteed to converge [Boyan, 2002].

## 2.3 iLSTD

iLSTD was recently introduced to provide a balance between LSTD's data efficiency and TD's time efficiency for $\lambda = 0$ when the feature representation is sparse [Geramifard *et al.*, 2006]. The basic idea is to maintain the same $\mathbf{A}$ matrix and $\mathbf{b}$ vector as LSTD, but to only incrementally solve for $\boldsymbol{\theta}$. The update to $\boldsymbol{\theta}$ requires some care as the sum TD update itself would require $O(n^2)$. iLSTD instead updates only single dimensions of $\boldsymbol{\theta}$, each of which requires $O(n)$. By updating $m$ parameters of $\boldsymbol{\theta}$, which is a parameter that can be varied to trade off data and computational efficiency, iLSTD requires $O(mn + k^2)$ per time step, which is linear in $n$. The result is that iLSTD can scale to much larger feature spaces than LSTD, while still retaining much of its data efficiency. Although the original formulation of iLSTD had no proof of convergence, it was shown in synthetic domains to perform nearly as well as LSTD with dramatically less computation.

In the remainder of the paper, we describe a generalization, iLSTD($\lambda$), of the original algorithm to handle $\lambda > 0$. By also generalizing the mechanism used to select the feature parameters to update, we additionally prove sufficient conditions for convergence.

## 3 The New Algorithm with Eligibility Traces

The iLSTD($\lambda$) algorithm is shown in Algorithm 1. The new algorithm is a generalization of the original iLSTD algorithm in two key ways. First, it uses eligibility traces ($\mathbf{z}$) to handle $\lambda > 0$. Line 5 updates $\mathbf{z}$, and lines 5–9 incrementally compute the same $\mathbf{A}_t$, $\mathbf{b}_t$, and $\boldsymbol{\mu}_t$ as described in Equation 1. Second, the dimension selection mechanism has been relaxed. Any feature selection mechanism can be employed in line 11 to select a dimension of the sum TD update vector ($\boldsymbol{\mu}$).[2] Line 12 will then take a step in that dimension, and line 13 updates the $\boldsymbol{\mu}$ vector accordingly. The original iLSTD algorithm can be recovered by simply setting $\lambda$ to zero and selecting features according to the dimension of $\boldsymbol{\mu}$ with maximal magnitude.

We now examine iLSTD($\lambda$)'s computational complexity.

| **Algorithm 1**: iLSTD($\lambda$) | Complexity |
|---|---|
| 0   $s \leftarrow s_0, \mathbf{z} \leftarrow \mathbf{0}, \mathbf{A} \leftarrow \mathbf{0}, \boldsymbol{\mu} \leftarrow \mathbf{0}, t \leftarrow 0$ | |
| 1   Initialize $\boldsymbol{\theta}$ arbitrarily | |
| 2   **repeat** | |
| 3     Take action according to $\pi$ and observe $r, s'$ | |
| 4     $t \leftarrow t + 1$ | |
| 5     $\mathbf{z} \leftarrow \gamma \lambda \mathbf{z} + \phi(s)$ | $O(n)$ |
| 6     $\Delta \mathbf{b} \leftarrow \mathbf{z} r$ | $O(n)$ |
| 7     $\Delta \mathbf{A} \leftarrow \mathbf{z}(\phi(s) - \gamma \phi(s'))^T$ | $O(kn)$ |
| 8     $\mathbf{A} \leftarrow \mathbf{A} + \Delta \mathbf{A}$ | $O(kn)$ |
| 9     $\boldsymbol{\mu} \leftarrow \boldsymbol{\mu} + \Delta \mathbf{b} - (\Delta \mathbf{A})\boldsymbol{\theta}$ | $O(kn)$ |
| 10    **for** $i$ from 1 to m **do** | |
| 11      $j \leftarrow$ choose an index of $\boldsymbol{\mu}$ using some *feature selection mechanism* | |
| 12      $\theta_j \leftarrow \theta_j + \alpha \mu_j$ | $O(1)$ |
| 13      $\boldsymbol{\mu} \leftarrow \boldsymbol{\mu} - \alpha \mu_j \mathbf{A} e_j$ | $O(n)$ |
| 14    **end for** | |
| 15    $s \leftarrow s'$ | |
| 16   **end repeat** | |

**Theorem 1** *Assume that the feature selection mechanism takes $O(n)$ computation. If there are $n$ features and, for any given state $s$, $\phi(s)$ has at most $k$ non-zero elements, then the iLSTD($\lambda$) algorithm requires $O((m + k)n)$ computation per time step.*

**Proof** Outside of the inner loop, lines 7–9 are the most computationally expensive steps of iLSTD($\lambda$). Since we assumed that each feature vector has at most $k$ non-zero elements, and the $\mathbf{z}$ vector can have up to $n$ non-zero elements, the $\mathbf{z}(\phi(s) - \gamma\phi(s'))^T$ matrix (line 7) has at most $2kn$ non-zero elements. This leads to $O(nk)$ complexity for the outside of the loop. Inside, the complexity remains unchanged from iLSTD with the most expensive lines being 11 and 13. Because $\boldsymbol{\mu}$ and $\mathbf{A}$ do not have any specific structure, the inside loop time[3] is $O(n)$. Thus, the final bound for the algorithm's per-time-step computational complexity is $O((m + k)n)$. $\qquad \square$

## 4   Convergence

We now consider the convergence properties of iLSTD($\lambda$). Our analysis follows that of Bertsekas and Tsitsiklis [1996] very closely to establish that iLSTD($\lambda$) converges to the same solution that TD($\lambda$) does. However, whereas in their analysis they considered $\mathbf{C}_t$ and $\mathbf{d}_t$ that had expectations that converged quickly, we consider $\mathbf{C}_t$ and $\mathbf{d}_t$ that may converge more slowly, but in value instead of expectation.

In order to establish our result, we consider the theoretical model where for all $t$, $\mathbf{y}_t \in \mathcal{R}^n, \mathbf{d}_t \in \mathcal{R}^n$, $\mathbf{R}_t, \mathbf{C}_t \in \mathcal{R}^{n \times n}, \beta_t \in \mathcal{R}$, and:

$$\mathbf{y}_{t+1} = \mathbf{y}_t + \beta_t(\mathbf{R}_t)(\mathbf{C}_t\mathbf{y}_t + \mathbf{d}_t). \qquad (2)$$

On every round, $\mathbf{C}_t$ and $\mathbf{d}_t$ are selected first, followed by $\mathbf{R}_t$. Define $F_t$ to be the state of the algorithm on round $t$ before $\mathbf{R}_t$ is selected. $\mathbf{C}_t$ and $\mathbf{d}_t$ are sequences of random variables. In order to prove convergence of $\mathbf{y}_t$, we assume that there is a $\mathbf{C}^*$, $\mathbf{d}^*$, $v, \mu > 0$, and $M$ such that:

    A1. $\mathbf{C}^*$ is negative definite,
    A2. $\mathbf{C}_t$ converges to $\mathbf{C}^*$ with probability 1,
    A3. $\mathbf{d}_t$ converges to $\mathbf{d}^*$ with probability 1,
    A4. $\mathbf{E}[\mathbf{R}_t|F_t] = I$, and $\|\mathbf{R}_t\| \leq M$,
    A5. $\lim_{T \to \infty} \sum_{t=1}^{T} \beta_t = \infty$, and
    A6. $\beta_t < vt^{-\mu}$.

**Theorem 2** *Given the above assumptions,* $\mathbf{y}_t$ *converges to* $-(\mathbf{C}^*)^{-1}\mathbf{d}^*$ *with probability 1.*

The proof of this theorem is included in the additional material and will be made available as a companion technical report. Now we can map iLSTD($\lambda$) on to this mathematical model:

1. $\mathbf{y}_t = \boldsymbol{\theta}_t$,
2. $\beta_t = t\alpha/n$,
3. $\mathbf{C}_t = -\mathbf{A}_t/t$,
4. $\mathbf{d}_t = \mathbf{b}_t/t$, and
5. $\mathbf{R}_t$ is a matrix, where there is an $n$ on the diagonal in position $(k_t, k_t)$ (where $k_t$ is uniform random over the set $\{1,\ldots, n\}$ and i.i.d.) and zeroes everywhere else.

The final assumption defines the simplest possible feature selection mechanism sufficient for convergence, *viz.*, uniform random selection of features.

**Theorem 3** *If the Markov decision process is finite, iLSTD($\lambda$) with a uniform random feature selection mechanism converges to the same result as TD($\lambda$).*

Although this result is for uniform random selection, note that Theorem 2 outlines a broad range of possible mechanisms sufficient for convergence. However, the greedy selection of the original iLSTD algorithm does not meet these conditions, and so has no guarantee of convergence. As we will see in the next section, though, greedy selection performs quite well despite this lack of asymptotic guarantee. In summary, finding a good feature selection mechanism remains an open research question.

As a final aside, one can go beyond iLSTD($\lambda$) and consider the case where $\mathbf{R}_t = \mathbf{I}$, *i.e.*, we take a step in all directions at once on every round. This does not correspond to any feature selection mechanism and in fact requires $O(n^2)$ computation. However, we can examine this algorithm's rate of convergence. In particular we find it converges linearly fast to LSTD($\lambda$).

**Theorem 4** *If* $\mathbf{C}_t$ *is negative definite, for some* $\beta$ *dependent upon* $\mathbf{C}_t$, *if* $\mathbf{R}_t = \mathbf{I}$, *then there exists an* $\zeta \in (0,1)$ *such that for all* $\mathbf{y}_t$, *if* $\mathbf{y}_{t+1} = \mathbf{y}_t + \beta(\mathbf{C}_t\mathbf{y}_t + \mathbf{d}_t)$, *then* $\left\|\mathbf{y}_{t+1} + (\mathbf{C}_t)^{-1}\mathbf{d}_t\right\| < \zeta \left\|\mathbf{y}_t + (\mathbf{C}_t)^{-1}\mathbf{d}_t\right\|$.

This may explain why iLSTD($\lambda$)'s performance, despite only updating a single dimension, approaches LSTD($\lambda$) so quickly in the experimental results in the next section.

## 5  Empirical Results

We now examine the empirical performance of iLSTD($\lambda$). We first consider the simple problem introduced by Boyan [2002] and on which the original iLSTD was evaluated. We then explore the larger mountain car problem with a tile coding function approximator. In both problems, we compare TD($\lambda$), LSTD($\lambda$), and two variants of iLSTD($\lambda$). We evaluate both the random feature selection mechanism ("iLSTD-random"), which is guaranteed to converge,[4] as well as the original iLSTD feature selection rule ("iLSTD-greedy"), which is not. In both cases, the number of dimensions picked per iteration is $m = 1$. The step size ($\alpha$) used for both iLSTD($\lambda$) and TD($\lambda$) was of the same form as in Boyan's experiments, with a slightly faster decay rate in order to make it consistent with the proof's assumption.

$$\alpha_t = \alpha_0 \frac{N_0 + 1}{N_0 + \text{Episode\#}^{1.1}}$$

For the TD($\lambda$) and iLSTD($\lambda$) algorithms, the best $\alpha_0$ and $N_0$ have been selected through experimental search of the sets of $\alpha_0 \in \{0.01, 0.1, 1\}$ and $N_0 \in \{100, 100, 10^6\}$ for each domain and $\lambda$ value, which is also consistent with Boyan's original experiments.

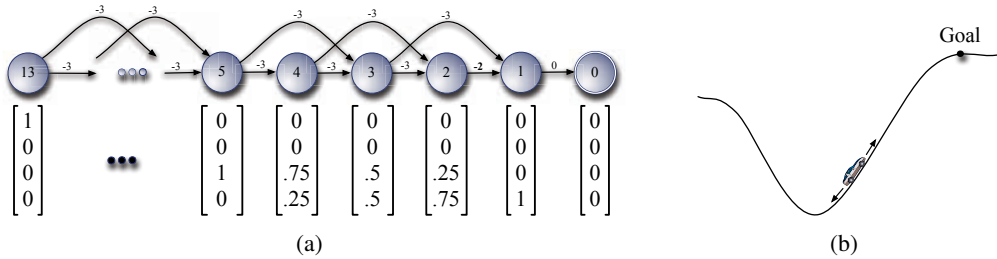

(a)                                                    (b)

Figure 1: The two experimental domains: (a) Boyan's chain example and (b) mountain car.

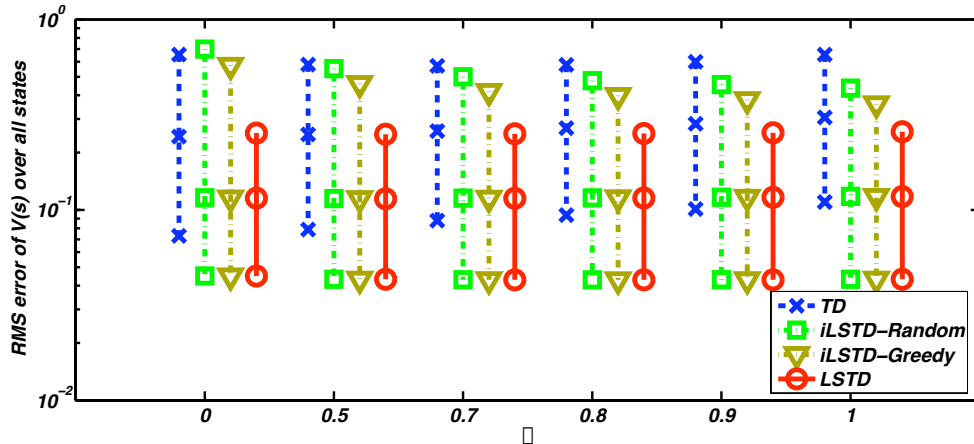

Figure 2: Performance of various algorithms in Boyan's chain problem with 6 different lambda values. Each line represents the averaged error over last 100 episodes after 100, 200, and 1000 episodes respectively. Results are also averaged over 30 trials.

## 5.1 Boyan Chain Problem

The first domain we consider is the Boyan chain problem. Figure 1(a) shows the Markov chain together with the feature vectors corresponding to each state. This is an episodic task where the discount factor $\gamma$ is one. The chain starts in state 13 and finishes in state 0. For all states $s > 2$, there exists an equal probability of ending up in $(s - 1)$ and $(s - 2)$. The reward is -3 for all transitions except from state 2 to 1 and state 1 to 0, where the rewards are -2 and 0, respectively.

Figure 2 shows the comparative results. The horizontal axis corresponds to different $\lambda$ values, while the vertical axis illustrates the RMS error in a log scale averaged over all states uniformly. Note that in this domain, the optimum solution is in the space spanned by the feature vectors: $\boldsymbol{\theta}^* = (-24, -16, -8, 0)^T$. Each line shows the averaged error over last 100 episodes after 100, 200, and 1000 episodes over the same set of observed trajectories based on 30 trials. As expected, LSTD($\lambda$) requires the least amount of data, obtaining a low average error after only 100 episodes. With only 200 episodes, though, the iLSTD($\lambda$) methods are performing as well as LSTD($\lambda$), and dramatically outperforming TD($\lambda$). Finally, notice that iLSTD-Greedy($\lambda$) despite its lack of asymptotic guarantee, is actually performing slightly better than iLSTD-Random($\lambda$) for all cases of $\lambda$. Although $\lambda$ did not play a significant role for LSTD($\lambda$) which matches the observation of Boyan [Boyan, 1999], $\lambda > 0$ does show an improvement in performance for the iLSTD($\lambda$) methods.

Table 1 shows the total averaged per-step CPU time for each method. For all methods sparse matrix optimizations were utilized and LSTD used the efficient incremental inverse implementation. Although TD($\lambda$) is the fastest method, the overall difference between the timings in this domain is very small, which is due to the small number of features and a small ratio $\frac{n}{k}$. In the next domain, we illustrate the effect of a larger and more interesting feature space where this ratio is larger.

|  | CPU time/step (msec) | |
| --- | --- | --- |
| Algorithm | Boyan's chain | Mountain car |
| TD($\lambda$) | 0.305±7.0e-4 | 5.35±3.5e-3 |
| iLSTD($\lambda$) | 0.370±7.0e-4 | 9.80±2.8e-1 |
| LSTD($\lambda$) | 0.367±7.0e-4 | 253.42 |

Table 1: The averaged CPU time per step of the algorithms used in Boyan's chain and mountain car problems.

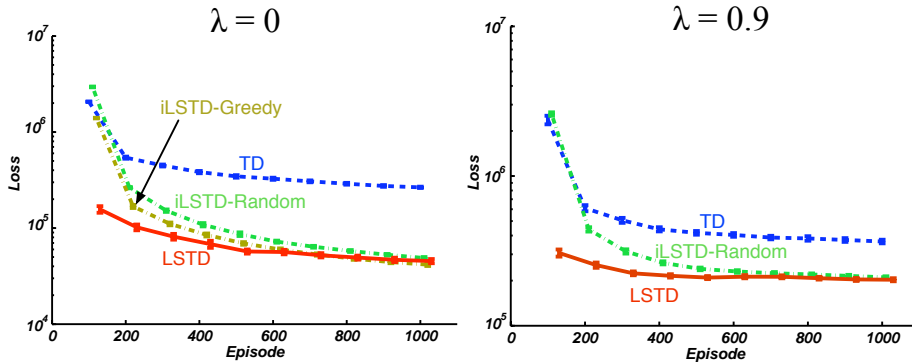

Figure 3: Performance of various methods in mountain car problem with two different lambda values. LSTD was run only every 100 episodes. Results are averaged over 30 trials.

## 5.2 Mountain Car

Our second test-bed is the mountain car domain [*e.g.*, see Sutton and Barto, 1998]. Illustrated in Figure 1(b), the episodic task for the car is to reach the goal state. Possible actions are accelerate forward, accelerate backward, and coast. The observation is a pair of continuous values: position and velocity. The initial value of the state was -1 for position and 0 for velocity. Further details about the mountain car problem are available online [RL Library, 2006]. As we are focusing on policy evaluation, the policy was fixed for the car to always accelerate in the direction of its current velocity, although the environment is stochastic and the chosen action is replaced with a random one with 10% probability. Tile Coding [*e.g.*, see Sutton, 1996] was selected as our linear function approximator. We used ten tilings ($k = 10$) over the combination of the two parameter sets and hashed the tilings into 10,000 features ($n = 10,000$). The rest of the settings were identical to those in the Boyan chain domain.

Figure 3 shows the results of the different methods on this problem with two different $\lambda$ values. The horizontal axis shows the number of episodes, while the vertical axis represents our loss function in log scale. The loss we used was $||\mathbf{b}^\lambda - \mathbf{A}^\lambda \boldsymbol{\theta}||_2$, where $\mathbf{A}^\lambda$ and $\mathbf{b}^\lambda$ were computed for each $\lambda$ from 200,000 episodes of interaction with the environment.

With $\lambda = 0$, both iLSTD($\lambda$) methods performed considerably better than TD($\lambda$) in terms of data efficiency. The iLSTD($\lambda$) methods even reached a level competitive with LSTD($\lambda$) after 600 episodes. For $\lambda = 0.9$, it proved to be difficult to find stable learning rate parameters for iLSTD-Greedy($\lambda$). While some iterations performed competitively with LSTD($\lambda$), others performed extremely poorly with little show of convergence. Hence, we did not include the performance line in the figure. This fact may suggest that the greedy feature selection mechanism does not converge, or it may simply be more sensitive to the learning rate. Finally, notice that the plotted loss depends upon $\lambda$, and so the two graphs cannot be directly compared.

In this environment the $\frac{n}{k}$ is relatively large ($\frac{10,000}{10} = 1000$), which translates into a dramatic improvement of iLSTD($\lambda$) over LSTD as can be see in Table 1. Again sparse matrix optimizations were utilized and LSTD($\lambda$) used the efficient incremental ivnerse implementation. The computational demands of LSTD($\lambda$) can easily prohibit its application in domains with a large feature space. When the feature representation is sparse, though, iLSTD($\lambda$) can still achieve results competitive with LSTD($\lambda$) using computation more on par with the time efficient TD($\lambda$).

## 6 Conclusion

In this paper, we extended the previous iLSTD algorithm by incorporating eligibility traces without increasing the asymptotic per time-step complexity. This extension resulted in improvements in performance in both the Boyan chain and mountain car domains. We also relaxed the dimension selection mechanism of the algorithm and presented sufficient conditions on the mechanism under which iLSTD($\lambda$) is guaranteed to converge. Our empirical results showed that while LSTD($\lambda$) can be impractical in on-line learning tasks with a large number of features, iLSTD($\lambda$) still scales well while having similar performance to LSTD.

This work opens up a number of interesting directions for future study. Our results have focused on two very simple feature selection mechanisms: random and greedy. Although the greedy mechanism does not meet our sufficient conditions for convergence, it actually performed slightly better on the examined domains than the theoretically guaranteed random selection. It would be interesting to perform a thorough exploration of possible mechanisms to find a mechanism with both good empirical performance while satisfying our sufficient conditions for convergence. In addition, it would be interesting to apply iLSTD($\lambda$) in even more challenging environments where the large number of features has completely prevented the least-squares approach, such as in simulated soccer keepaway [Stone *et al.*, 2005].

## Footnotes

[1]Throughout this paper we will use non-bolded symbols to refer to scalars (*e.g.*, $\gamma$ and $\alpha_t$), bold-faced lower-case symbols to refer to vectors (*e.g.*, $\boldsymbol{\theta}$ and $\mathbf{b}_t$), and bold-faced upper-case symbols for matrices (*e.g.*, $\mathbf{A}_t$).

[2]The choice of this mechanism will determine the convergence properties of the algorithm, as discussed in the next section.

[3] Note that $\mathbf{A}e_i$ selects the $i$th column of $\mathbf{A}$ and so does not require the usual quadratic time for multiplying a vector by a square matrix.

[4]When selecting features randomly we exclude dimensions with zero sum TD update. To be consistent with the assumptions of Theorem 2, we compensate by multiplying the learning rate $\alpha_t$ by the fraction of features that are non-zero at time $t$.

## References

[Bertsekas and Tsitsiklis, 1996] Dmitri P. Bertsekas and John N. Tsitsiklis. *Neuro-Dynamic Programming*. Athena Scientific, 1996.

[Boyan, 1999] Justin A. Boyan. Least-squares temporal difference learning. In *Proceedings of the Sixteenth International Conference on Machine Learning*, pages 49–56. Morgan Kaufmann, San Francisco, CA, 1999.

[Boyan, 2002] Justin A. Boyan. Technical update: Least-squares temporal difference learning. *Machine Learning*, 49:233–246, 2002.

[Bradtke and Barto, 1996] S. Bradtke and A. Barto. Linear least-squares algorithms for temporal difference learning. *Machine Learning*, 22:33–57, 1996.

[Geramifard *et al.*, 2006] Alborz Geramifard, Michael Bowling, and Richard S. Sutton. Incremental least-squares temporal difference learning. In *Proceedings of the Twenty-First National Conference on Artificial Intelligence (AAAI)*, pages 356–361. AAAI Press, 2006.

[RL Library, 2006] RL Library. The University of Alberta reinforcement learning library. `http://rlai.cs.ualberta.ca/RLR/environment.html`, 2006.

[Stone *et al.*, 2005] Peter Stone, Richard S. Sutton, and Gregory Kuhlmann. Reinforcement learning for robocup soccer keepaway. *International Society for Adaptive Behavior*, 13(3):165–188, 2005.

[Sutton and Barto, 1998] R. S. Sutton and A. G. Barto. *Reinforcement Learning: An Introduction*. MIT Press, 1998.

[Sutton, 1988] Richard S. Sutton. Learning to predict by the methods of temporal differences. *Machine Learning*, 3:9–44, 1988.

[Sutton, 1996] Richard S. Sutton. Generalization in reinforcement learning: Successful examples using sparse coarse coding. In *Advances in Neural Information Processing Systems 8*, pages 1038–1044. The MIT Press, 1996.

[Tsitsiklis and Van Roy, 1997] John N. Tsitsiklis and Benjamin Van Roy. An analysis of temporal-difference learning with function approximation. *IEEE Transactions on Automatic Control*, 42(5):674–690, 1997.
